# Forward-Backward Activation Algorithm for Hierarchical Hidden Markov Models

**Kei Wakabayashi**
Faculty of Library, Information and Media Science
University of Tsukuba, Japan
kwakaba@slis.tsukuba.ac.jp

**Takao Miura**
Department of Engineering
Hosei University, Japan
miurat@hosei.ac.jp

## Abstract

Hierarchical Hidden Markov Models (HHMMs) are sophisticated stochastic models that enable us to capture a hierarchical context characterization of sequence data. However, existing HHMM parameter estimation methods require large computations of time complexity $O(TN^{2D})$ at least for model inference, where $D$ is the depth of the hierarchy, $N$ is the number of states in each level, and $T$ is the sequence length. In this paper, we propose a new inference method of HHMMs for which the time complexity is $O(TN^{D+1})$. A key idea of our algorithm is application of the forward-backward algorithm to *state activation probabilities*. The notion of a state activation, which offers a simple formalization of the hierarchical transition behavior of HHMMs, enables us to conduct model inference efficiently. We present some experiments to demonstrate that our proposed method works more efficiently to estimate HHMM parameters than do some existing methods such as the flattening method and Gibbs sampling method.

## 1 Introduction

Latent structure analysis of sequence data is an important technique for many applications such as speech recognition, bioinformatics, and natural language processing. Hidden Markov Models (HMMs) play a key role in solving these problems. HMMs assume a single Markov chain of hidden states as the latent structure of sequence data. Because of this simple assumption, HMMs tend to capture only local context patterns of sequence data. Hierarchical Hidden Markov Models (HH-MMs) are stochastic models which assume hierarchical Markov chains of hidden states as the latent structure of sequence data [3]. HHMMs have a hierarchical state transition mechanism that yields the capability of capturing global and local sequence patterns in various granularities. By their nature, HHMMs are applicable to problems of many kinds including handwritten letter recognition [3], information extraction from documents [11], musical pitch structure modeling [12], video structure modeling [13], and human activity modeling [8, 6].

For conventional HMMs, we can conduct unsupervised learning efficiently using the *forward-backward algorithm*, which is a kind of dynamic programming [9]. In situations where few or no supervised data are available, the existence of the efficient unsupervised learning algorithm is a salient advantage of using HMMs. The unsupervised learning of HHMMs is an important technique, as it is for HMMs. In this paper, we discuss unsupervised learning techniques for HHMMs. We introduce a key notion, *activation probability*, to formalize the hierarchical transition mechanism naturally. Using this notion, we propose a new exact inference algorithm which has less time complexity than existing methods have.

The remainder of the paper is organized as follows. In section 2, we overview HHMMs. In section 3, we survey HHMM parameter estimation techniques proposed to date. In section 4, we introduce our parameter estimation algorithm. Section 5 presents experiments to show the effectiveness of our algorithm. We conclude our discussion in section 6.

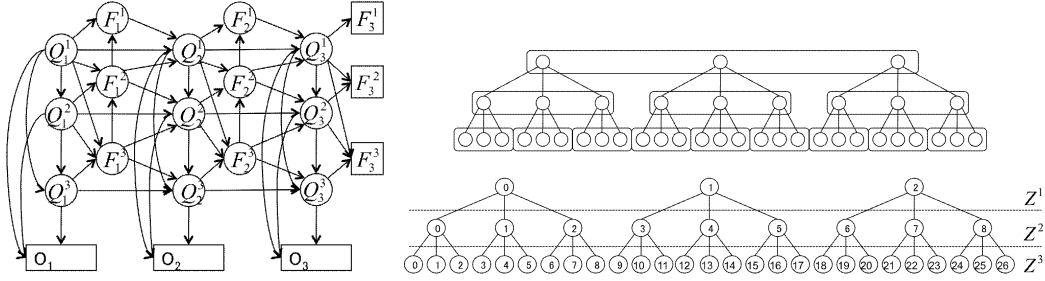

Figure 1: **(left)** Dynamic Bayesian network of the HHMM. **(top-right)** Tree representation of the HHMM state space. **(bottom-right)** State identification by the absolute path of the tree.

## 2 Hierarchical Hidden Markov Models

Let $O = \{O_1, ..., O_t, ..., O_T\}$ be a sequence of observations in which subscript $t$ denotes the time in the sequence. We designate *time* as an integer index of observation numbered from the beginning of the sequence. HHMMs define $Q_t^d$ for $1 \leq t \leq T, 1 \leq d \leq D$ as a hidden state at time $t$ and level $d$, where $d = 1$ represents the top level and $d = D$ represents the bottom level. HHMMs also define binary variables $F_t^d$, called *termination indicators*. If $F_t^d = 1$, then it is indicated that the Markov chain of level $d$ terminates at time $t$. In HHMMs, a state transition at level $d$ is permitted only when the Markov chain of level $d + 1$ terminates, i.e. $Q_t^d = Q_{t-1}^d$ if $F_{t-1}^{d+1} = 0$. A terminated Markov chain is initialized again at the next time. Figure 1 (left) presents a Dynamic Bayesian Network (DBN) expression for an HHMM of hierarchical depth $D = 3$. The conditional probability distribution of $Q$, $F$ and $O$ is defined as follows [7].

$$p(Q_t^d = j | Q_{t-1}^d = i, F_{t-1}^{d+1} = b, F_{t-1}^d = f, Q_t^{1:d-1} = k) = \begin{cases} \delta(i,j) & (\text{if } b = 0) \\ A_k^d(i,j) & (\text{if } b = 1, f = 0) \\ \pi_k^d(j) & (\text{if } b = 1, f = 1) \end{cases}$$

$$p(F_t^d = 1 | Q_t^d = i, Q_t^{1:d-1} = k, F_t^{d+1} = b) = \begin{cases} 0 & (\text{if } b = 0) \\ A_k^d(i, end) & (\text{if } b = 1) \end{cases}$$

$$p(O_t = v | Q_t^{1:D} = k) = B_k(v)$$

We use a notation $Q_t^{1:d-1}$ as a combination of states $\{Q_t^1, ..., Q_t^{d-1}\}$. Probabilities of the initialization and the state transition of Markov chains at level $d$ depend on all higher states $Q^{1:d-1}$. $A_k^d(i,j)$ is a model parameter of the transition probability at level $d$ from state $i$ to $j$ when $Q_t^{1:d-1} = k$. $A_k^d(i, end)$ denotes a termination probability that state $i$ terminates the Markov chain at level $d$ when $Q_t^{1:d-1} = k$. $\pi_k^d(j)$ is an initial state probability of state $j$ at level $d$ when $Q_t^{1:d-1} = k$. $B_k(v)$ is an output probability of observation $v$ when $Q_t^{1:D} = k$.

A state space of HHMM is expressed as a tree structure [3]. Figure 1 (top-right) presents a tree expression of state space of an HHMM for which the depth $D = 3$ and the number of states in each level $N = 3$. The level of the tree corresponds to the level of HHMM states. Each node at level $d$ corresponds to a combination of states $Q^{1:d}$. Each node has $N$ children because there are $N$ possible states for each level. The rectangles in the figure denote local HMMs in which nodes can mutually transit directly using the transition probability $A$. For the analysis described herein, we assume the balanced N-ary tree to simplify discussions of computational complexity. However, arbitrary state space trees do not change the substance of what follows.

The behavior of Markov chain at level $d$ depends on the combination of all higher-up states $Q^{1:d-1}$, not only on the individual $Q^d$. In the tree structure, the absolute path which corresponds to $Q^{1:d}$ is meaningful, rather than the relative path which corresponds to $Q^d$. We refer to $Q^{1:d}$ as $Z^d$ and call it *absolute path state*. Figure 1 (bottom-right) presents an absolute path state identification. The set of values taken by an absolute path state at level $d$, denoted by $\Omega^d$, contains $N^d$ elements in the balanced N-ary tree state space. We define a function to obtain the parent absolute path state of $Z^d$ as $parent(Z^d)$. Similarly, we define a function to obtain the set of child absolute path states of $Z^d$ as $child(Z^d)$, and a function to obtain the set of siblings of $Z^d$ as $sib(Z^d) = child(parent(Z^d))$.

Table 1: Notation for HHMMs.

| | |
|---|---|
| $D$ | Depth of hierarchy |
| $N$ | Number of states in each level |
| $\Omega^d$ | Set of values taken by absolute path state at level $d$ |
| $Z_t^d \in \Omega^d$ | Absolute path state at time $t$ and level $d$ |
| $F_t^d \in \{0, 1\}$ | Termination indicator at time $t$ and level $d$ |
| $O_t \in \{1, ..., V\}$ | Observation at time $t$ |
| $A_{dij}$ | State transition probability from state $Z_t^d = i$ to state $Z_{t+1}^d = j$ at level $d$ |
| $A_{diEnd}$ | Termination probability of Markov chain at level $d$ from state $Z_t^d = i$ |
| $\pi_{di}$ | Initial state probability of state $Z^d = i$ at level $d$ |
| $B_{iv}$ | Output probability of observation $v$ with $Z^D = i$ |

Table 1 presents the notation used for the HHMM description. We use the notation of the absolute path state $Z^d$ rather than $Q^d$ throughout the paper. Therefore, we define compatible notations for the model parameters. Whereas the conventional notation $\pi_k^d(j)$ denotes the initial state probability of $Q^d = j$ when $Q^{1:d-1} = k$, we aggregate $Q^d$ and $Q^{1:d-1}$ into $Q^{1:d} = Z^d$ and define $\pi_{di}$ as the initial state probability of $Z^d = i$. Similarly, we define $A_{dij}$ as the state transition probability from $Z^d = i$ to $j$. Note that $\sum_{i' \in sib(i)} \pi_{di'} = 1$ and $\sum_{j' \in \{sib(i) \cup End\}} A_{dij'} = 1$.

## 3 Existing Parameter Estimation Methods for HHMMs

The first work for HHMMs [3] proposed the *generalized Baum-Welch algorithm*. This algorithm is based on an inside-outside algorithm used for inference of probabilistic context free grammars. This method takes $O(T^3)$ time complexity, which is not practical for long sequence data.

A more efficient approach is the *flattening method* [7]. The hierarchical state sequence can be reduced to a single sequence of the bottom level absolute path states $\{Z_1^D, ..., Z_T^D\}$. If we regard $Z^D$ as a *flat* HMM state, then we can conduct the inference by using the forward-backward algorithm with $O(TN^{2D})$ time complexity since $|\Omega^D| = N^D$. Notice that the flat state $Z^D$ can transit to any other flat state, and we cannot apply efficient algorithms for HMMs of sparse transition matrix. In the flattening method, we must make a weak constraint on the HHMM parameters, say *minimally self-referential (MinSR)* [12], which restricts the self-transition at higher levels i.e. $A_{dii} = 0$ for $1 \leq d \leq D-1$. The MinSR constraint enables us to identify the path connecting two flat states uniquely. This property is necessary for estimating HHMM parameters by using the flattening method.

We also discuss a sampling approach as an alternative parameter estimation technique. The Gibbs sampling is often used for parameter estimation of probabilistic models including latent variables [4]. We can estimate HMM parameters using a Gibbs sampler, which sample each hidden states iteratively. This method is applicable to inference of HHMMs in a straightforward manner on the flat HMM. This straightforward approach, called the Direct Gibbs Sampler (DGS), takes the $O(TN^D)$ time complexity for a single iteration.

The convergence of a posterior distribution by the DGS method is said to be extremely slow for HMMs [10] because the DGS ignores *long time dependencies*. Chib [2] introduced an alternative method, called the Forward-Backward Gibbs Sampler (FBS), which calculates forward probabilities in advance. FBS samples hidden states from the end of the sequence regarding the forward probabilities. FBS method requires larger computations for a single iteration than DGS does, but it can bring a posterior of hidden states to its stationary distribution with fewer iterations [10].

Heller [5] proposed *Infinite Hierarchical Hidden Markov Models* (IHHMMs) which can have an infinitely large depth by weakening the dependency between the states at different levels. They proposed the inference method for IHHMMs based on a blocked Gibbs sampler of which the sampling unit is a state sequence from $t = 1$ to $T$ at a single level. This inference takes only $O(TD)$ time for a single iteration. In HHMMs, the states in each level are strongly dependent, so resampling a state at an intermediate level causes all lower states to alter into a state which has a completely different behavior. Therefore, it is not practical to apply this Gibbs sampler to HHMMs in terms of the convergence speed.

## 4 Forward-Backward Activation Algorithm

In this section, we introduce a new parameter estimation algorithm for HHMMs, which theoretically has $O(TN^{D+1})$ time complexity. The basic idea of our algorithm is a decomposition of the flat transition probability distribution $p(Z_{t+1}^D|Z_t^D)$, which the flattening method calculates directly for all pairs of the flat states. We can rewrite the flat transition probability distribution into a sum of two cases that the Markov chain at level $D$ terminates or not, as follows.

$$
\begin{aligned}
p(Z_{t+1}^D|Z_t^D) \;=\; & p(Z_{t+1}^D|Z_t^D, F_t^D = 0)p(F_t^D = 0|Z_t^D) + \\
& p(Z_{t+1}^D|Z_{t+1}^{D-1}, F_t^D = 1)p(Z_{t+1}^{D-1}|Z_t^{D-1}, F_t^D = 1)p(F_t^D = 1|Z_t^D)
\end{aligned}
$$

The first term corresponds to the direct transition without the Markov chain termination. The actual computational complexity for calculating this term is $O(N^{D+1})$ because the direct transition is permitted only between the sibling states, i.e. $A_{Dij} = 0$ if $j \notin sib(i)$. The second term, corresponding to the case in which the Markov chain terminates at level $D$, contains two factors: The upper level transition probability $p(Z_{t+1}^{D-1}|Z_t^{D-1}, F_t^D = 1)$ and the state initialization probability for the terminated Markov chain $p(Z_{t+1}^D|Z_{t+1}^{D-1}, F_t^D = 1)$. We attempt to compute these probability distributions efficiently in a dynamic programming manner.

The transition probability at level $d$ has the form $p(Z_{t+1}^d|Z_t^d, F_t^{d+1} = 1)$. We define *ending activation* $e_t^d$, as the condition of the transition probability from $Z_t^d$, formally:

$$
p(e_t^d = i) = \left\{
\begin{array}{ll}
p(Z_t^d = i, F_t^{d+1} = 1) & (\texttt{if } i \neq null \texttt{ and } d < D) \\
p(Z_t^d = i) & (\texttt{if } i \neq null \texttt{ and } d = D) \\
p(F_t^{d+1} = 0) & (\texttt{if } i = null)
\end{array}
\right.
$$

The null value in $e_t^d$ indicates that the Markov chain at level $d + 1$ does not terminate at time $t$. The state initialization probability for level $d + 1$ has the form $p(Z_t^{d+1}|Z_t^d, F_{t-1}^{d+1} = 1)$. We define *beginning activation* $b_t^d$, as the condition of the state initialization probability from $Z_t^d$, formally, as

$$
p(b_t^d = i) = \left\{
\begin{array}{ll}
p(Z_t^d = i, F_{t-1}^{d+1} = 1) & (\texttt{if } i \neq null \texttt{ and } d < D \texttt{ and } t > 1) \\
p(Z_t^d = i) & (\texttt{if } i \neq null \texttt{ and } (d = D \texttt{ or } t = 1)) \\
p(F_{t-1}^{d+1} = 0) & (\texttt{if } i = null)
\end{array}
\right.
$$

The null value in $b_t^d$ indicates that the Markov chain at level $d + 1$ does not terminate at time $t - 1$.

Using these notations, we can represent the flat transition with propagations of activation probabilities as shown in figure 2 (left) because $p(Z_{t+1}^D|Z_t^D) = p(b_{t+1}^D|e_t^D)$. This representation naturally describes the decomposition of the flat transition probability distribution discussed above, and it enables us to apply the decomposition recursively for all levels. We can derive the conditional probability distributions of $e_t^d$ and $b_{t+1}^d$ as

$$
p(e_t^d = i|e_t^{d+1}) = \left\{
\begin{array}{ll}
\sum_{c \in child(i)} p(e_t^{d+1} = c)A_{(d+1)cEnd} & (\texttt{if } i \neq null) \\
\sum_{c \in \Omega^{d+1}} p(e_t^{d+1} = c)(1 - A_{(d+1)cEnd}) + p(e_t^{d+1} = null) & (\texttt{if } i = null)
\end{array}
\right.
$$

$$
p(b_{t+1}^d = i|e_t^d, b_{t+1}^{d-1}) = \left\{
\begin{array}{ll}
p(b_{t+1}^{d-1} = parent(i))\pi_{di} + \sum_{j \in sib(i)} p(e_t^d = j)A_{dji} & (\texttt{if } i \neq null) \\
p(e_t^d = null) & (\texttt{if } i = null)
\end{array}
\right.
$$

In the following subsections, we show the efficient inference algorithm and the parameter estimation algorithm using the activation probabilities.

### 4.1 Inference using Forward and Backward Activation Probabilities

We can translate the DBN of HHMMs in figure 1 (left) equivalently into simpler DBN using activation probabilities. The translated DBN is portrayed in figure 2 (right). The inference algorithm proposed herein is based on a forward-backward calculation over this DBN. We define *forward activation probability* $\alpha$ and *backward activation probability* $\beta$ as follows.

$$
\begin{aligned}
\alpha_{e_t^d}(i) &= p(e_t^d = i, O_{1:t}) \\
\alpha_{b_t^d}(i) &= p(b_t^d = i, O_{1:t-1}) \\
\beta_{e_t^d}(i) &= p(O_{t+1:T}, F_T^1 = 1|e_t^d = i) \\
\beta_{b_t^d}(i) &= p(O_{t:T}, F_T^1 = 1|b_t^d = i)
\end{aligned}
$$

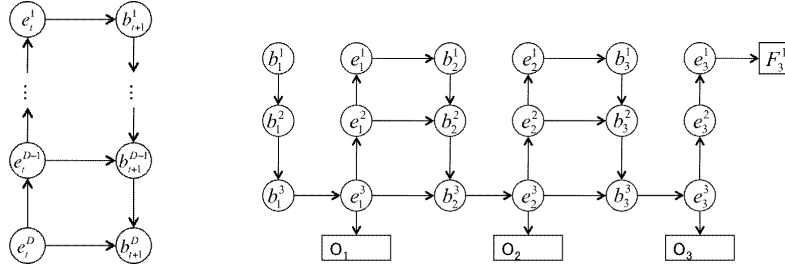

Figure 2: **(left)** Propagation of activation probabilities for calculating the flat transition probability from time $t$ to $t+1$. **(right)** Equivalent DBN of the HHMM using activation probabilities.

---

**Algorithm 1** Calculate forward activation probabilities

---
1: **for** $t = 1$ to $T$ **do**
2:    **if** $t = 1$ **then**
3:       $\alpha_{b_1^1}(i \in \Omega^1) = \pi_{1i}$
4:       **for** $d = 2$ to $D$ **do**
5:          $\alpha_{b_1^d}(i \in \Omega^d) = \alpha_{b_1^{d-1}}(parent(i))\pi_{di}$
6:       **end for**
7:    **else**
8:       $\alpha_{b_t^1}(i \in \Omega^1) = \sum_{j \in sib(i)} \alpha_{e_{t-1}^1}(j) A_{1ji}$
9:       **for** $d = 2$ to $D$ **do**
10:         $\alpha_{b_t^d}(i \in \Omega^d) = \alpha_{b_t^{d-1}}(parent(i))\pi_{di} + \sum_{j \in sib(i)} \alpha_{e_{t-1}^d}(j) A_{dji}$
11:       **end for**
12:    **end if**
13:    $\alpha_{e_t^D}(i \in \Omega^D) = \alpha_{b_t^D}(i) B_{iO_t}$
14:    **for** $d = D - 1$ to $1$ **do**
15:       $\alpha_{e_t^d}(i \in \Omega^d) = \sum_{c \in child(i)} \alpha_{e_t^{d+1}}(c) A_{(d+1)cEnd}$
16:    **end for**
17: **end for**

---

These probabilities are efficiently calculable in a dynamic programming manner. Algorithm 1 presents the pseudocodes to calculate whole $\alpha$. $\alpha_{b_t^d}$ are derived downward from $\alpha_{b_t^1}$ to $\alpha_{b_t^D}$ by summing up to the initialization probability from the parent and the transition probabilities from the siblings (Line 8 to 11). $\alpha_{e_t^d}$ are propagated upward from $\alpha_{e_t^D}$ to $\alpha_{e_t^1}$ by summing up to the probabilities of the child Markov chain termination (Line 13 to 16). This algorithm includes the calculation of $|\Omega^d| = N^d$ quantities involving the summation of $|sib(i)| = N$ terms for $d = 1$ to $D$ and for $t = 1$ to $T$. Therefore, the time complexity of algorithm 1 is $O(T \sum_{d=1}^{D} N^{d+1}) = O(TN^{D+1})$. Algorithm 2 propagates the backward activation probabilities similarly in backward order.

We can derive the conditional independence of $O_{1:t}$ and $\{O_{t+1:T}, F_T^1 = 1\}$ given $e_t^d \neq null$ or $b_{t+1}^d \neq null$, because both of $e_t^d \neq null$ and $b_{t+1}^d \neq null$ indicates that the Markov chains at level $d + 1, ..., D$ terminates at time $t$. On the basis of this conditional independence, the exact inference of a posterior of activation probabilities can be obtained using $\alpha$ and $\beta$ as presented below.

$$p(e_t^d = i | O_{1:T}, F_T^1 = 1) \propto p(e_t^d = i, O_{1:t}) p(O_{t+1:T}, F_T^1 = 1 | e_t^d = i) = \alpha_{e_t^d}(i) \beta_{e_t^d}(i)$$

$$p(b_t^d = i | O_{1:T}, F_T^1 = 1) \propto p(b_t^d = i, O_{1:t-1}) p(O_{t:T}, F_T^1 = 1 | b_t^d = i) = \alpha_{b_t^d}(i) \beta_{b_t^d}(i)$$

The inference of the flat state $p(Z_t^D | O_{1:T}, F_T^1 = 1)$ is identical to of the bottom level activation probability $p(e_t^D | O_{1:T}, F_T^1 = 1)$. We can calculate the likelihood of the whole observation as follows.

$$p(O_{1:T}, F_T^1 = 1) = \sum_{i \in \Omega^1} p(e_T^1 = i, O_{1:T}) p(F_T^1 = 1 | e_T^1 = i) = \sum_{i \in \Omega^1} \alpha_{e_T^1}(i) \beta_{e_T^1}(i)$$

**Algorithm 2** Calculate backward activation probabilities

1: **for** $t = T$ to $1$ **do**
2:   **if** $t = T$ **then**
3:     $\beta_{e_T^1}(i \in \Omega^1) = A_{1iEnd}$
4:     **for** $d = 2$ to $D$ **do**
5:       $\beta_{e_T^d}(i \in \Omega^d) = \beta_{e_T^{d-1}}(parent(i))A_{diEnd}$
6:     **end for**
7:   **else**
8:     $\beta_{e_t^1}(i \in \Omega^1) = \sum_{j \in sib(i)} \beta_{b_{t+1}^1}(j)A_{1ij}$
9:     **for** $d = 2$ to $D$ **do**
10:       $\beta_{e_t^d}(i \in \Omega^d) = \beta_{e_t^{d-1}}(parent(i))A_{diEnd} + \sum_{j \in sib(i)}\beta_{b_{t+1}^d}(j)A_{dij}$
11:     **end for**
12:   **end if**
13:   $\beta_{b_t^D}(i \in \Omega^D) = \beta_{e_t^D}(i)B_{iO_t}$
14:   **for** $d = D - 1$ to $1$ **do**
15:     $\beta_{b_t^d}(i \in \Omega^d) = \sum_{c \in child(i)} \beta_{b_t^{d+1}}(c)\pi_{(d+1)c}$
16:   **end for**
17: **end for**

## 4.2 Updating Parameters

Using the forward and backward activation probabilities, we can estimate HHMM parameters efficiently in the EM framework. In the EM algorithm, the function $Q(\theta, \bar{\theta})$ is defined, where $\theta$ is a parameter set before updating and $\bar{\theta}$ is a parameter set after updating, as described below.

$$Q(\theta, \bar{\theta}) = \sum_Y p_\theta(Y|X) \log p_{\bar{\theta}}(X, Y)$$

In that equation, $X$ represents a set of observed variables, and $Y$ is a set of latent variables. The difference of log likelihood between the models of $\theta$ and $\bar{\theta}$ is known to be greater than $Q(\theta, \bar{\theta}) - Q(\theta, \theta)$ [1]. For this reason, we can increase the likelihood monotonically by selecting a new parameter $\hat{\theta}$ to maximize the function $Q$. For HHMMs, the set of parameters is $\theta = \{A, \pi, B\}$. The set of observed variables is $X = \{O_{1:T}, F_T^1 = 1\}$. The set of latent variables is $Y = \{Z_{1:T}^{1:D}, F_{1:T-1}^{1:D}\}$. Therefore, the function $Q$ can be represented as shown below.

$$Q(\theta, \bar{\theta}) \propto \sum_{Z_{1:T}^{1:D}, F_{1:T-1}^{1:D}} p_\theta(O_{1:T}, F_T^1 = 1, Z_{1:T}^{1:D}, F_{1:T-1}^{1:D}) \log p_{\bar{\theta}}(O_{1:T}, F_T^1 = 1, Z_{1:T}^{1:D}, F_{1:T-1}^{1:D}) \quad (1)$$

The joint probability of observed variables and latent variables is given below.

$$p_\theta(O_{1:T}, F_T^1 = 1, Z_{1:T}^{1:D}, F_{1:T-1}^{1:D})$$
$$= \prod_{d=1}^{D} \pi_{dZ_1^d} \prod_{t=1}^{T-1} \prod_{d=1}^{D} (A_{dZ_t^d End}^{F_t^d} A_{dZ_t^d Z_{t+1}^d}^{F_t^{d+1}(1-F_t^d)} \pi_{dZ_{t+1}^d}^{F_t^d}) \prod_{d=1}^{D} A_{dZ_T^d End} \prod_{t=1}^{T} B_{Z_t^D O_t}$$

We substitute this equation for the joint probability in equation (1). We integrate out irrelevant variables and organize around each parameter. Thereby, we obtain the following.

$$Q(\theta, \bar{\theta}) \propto \sum_{d=1}^{D} \sum_{i \in \Omega^d} g_{\pi di} \log \bar{\pi}_{di} + \sum_{d=1}^{D} \sum_{i \in \Omega^d} \sum_{j \in \{sib(i) \cup End\}} g_{Adij} \log \bar{A}_{dij} + \sum_{i \in \Omega^D} \sum_{v=1}^{V} g_{Biv} \log \bar{B}_{iv}$$

Therein, $g_{\pi di}$, $g_{Adij}$, $g_{Biv}$ are shown by equation (2)(3)(4)(5). They are calculable using forward and backward activation probabilities.

$$g_{\pi di} = \alpha_{b_1^d}(i)\beta_{b_1^d}(i) + \sum_{t=1}^{T-1} \alpha_{b_{t+1}^{d-1}}(parent(i))\pi_{di}\beta_{b_{t+1}^d}(i) \quad (2)$$

$$g_{AdiEnd} = \sum_{t=1}^{T-1} \alpha_{e_t^d}(i)A_{diEnd}\beta_{e_t^{d-1}}(parent(i)) + \alpha_{e_T^d}(i)\beta_{e_T^d}(i) \quad (3)$$

Table 2: Log-likelihood achieved at each iteration.

| Iteration | 1 | 2 | 3 | 4 | 5 | 10 | 50 | 100 |
|---|---|---|---|---|---|---|---|---|
| FBA w/o MinSR | -773.47 | -672.44 | -668.50 | -631.30 | -610.63 | -577.33 | -457.66 | -447.90 |
| FBA with MinSR | -773.89 | -672.47 | -670.40 | -643.62 | -614.98 | -573.84 | -453.09 | -448.52 |
| FFB | -773.89 | -672.47 | -670.40 | -643.62 | -614.98 | -573.84 | -453.09 | -448.52 |

$$g_{Adij} = \sum_{t=1}^{T-1} \alpha_{e_t^d}(i) A_{dij} \beta_{b_{t+1}^d}(j) \tag{4}$$

$$g_{Biv} = \sum_{t:O_t=v} \alpha_{e_t^D}(i) \beta_{e_t^D}(i) \tag{5}$$

Using Lagrange multipliers, we can obtain parameters $\bar{\pi}, \bar{A}, \bar{B}$, which maximize the function $Q$ under the constraint $\sum_{i' \in sib(i)} \bar{\pi}_{di'} = 1, \sum_{j' \in \{sib(i) \cup End\}} \bar{A}_{dij'} = 1, \sum_v \bar{B}_{iv} = 1$ as shown below.

$$\bar{\pi}_{di} = \frac{g_{\pi di}}{\sum_{i' \in sib(i)} g_{\pi di'}}, \bar{A}_{dij} = \frac{g_{Adij}}{\sum_{j' \in \{sib(i) \cup End\}} g_{Adij'}}, \bar{B}_{iv} = \frac{g_{Biv}}{\sum_v g_{Biv}}$$

Consequently, we can calculate the update parameters using $\alpha$ and $\beta$. The time complexity for computing a single EM iteration is $O(TN^{D+1})$, which is identical to the calculation of forward and backward activation probabilities.

## 5  Experiments

Firstly, we experimentally confirm that the forward-backward activation algorithm yields exactly identical parameter estimation to the flattening method does. Remind that we must make the MinSR constraint on the HHMM parameter set in the flattening method (see section 3). We compare three parameter estimation algorithms: our forward-backward activation algorithm for a MinSR HHMM (FBA with MinSR), for a HHMM without MinSR (FBA w/o MinSR), and the flattening method(FFB). The dataset to learn includes 5 sequences of 10 length, which are artificially generated by a MinSR HHMM of biased parameter set. We execute three algorithms and examine the log-likelihood achieved at each iteration.

Table 2 presents the result. The FBA with MinSR and the FFB achieve the identical log-likelihood through the training. This result provides experimental evidence that our algorithm estimates HHMM parameters exactly identically to the flattening method does. Furthermore, the FBA enables us to conduct the parameter estimation of HHMMs which has non-zero self-transition parameters.

To evaluate the computational costs empirically, we compare four methods of HHMM parameter estimation. Two are based on the EM algorithm with inference by the forward-backward activation algorithm (FBA), and by the flattening forward-backward method (FFB). Another two are based on a sampling approach: direct Gibbs sampling for the flat HMMs (DGS) and forward-backward activation sampling (FBAS). FBAS is a straightforward application of the forward-backward sampling scheme to the translated DBN presented in figure 2. In FBAS, we first calculate forward activation probabilities. Then we sample state activation variables from $e_T^1$ to $b_1^1$ in the backward order with respect to forward activation probabilities. We evaluate four methods based on three aspects: execution time, convergence speed, and scalability of the state space size. We apply each method to four different HHMMs of $(D = 3, N = 3)$, $(D = 3, N = 4)$, $(D = 4, N = 3)$, and $(D = 4, N = 4)$. We examine the log-likelihood of the training dataset achieved at each iteration to ascertain the learning convergence. As a training dataset, we use 100 documents from the Reuters corpus as word sequences. The dataset includes 36,262 words in all, with a 4,899 word vocabulary.

Figure 3 presents the log-likelihood of the training data. The horizontal axis shows the logarithmically scaled execution time. Table 2 presents the average execution time for a single iteration. From these results, we can say primarily that FBA outperforms FFB in terms of execution time. The improvement is remarkable, especially for the HHMMs of large state space size because FBA has less time complexity for $N$ and $D$ than FFB has.

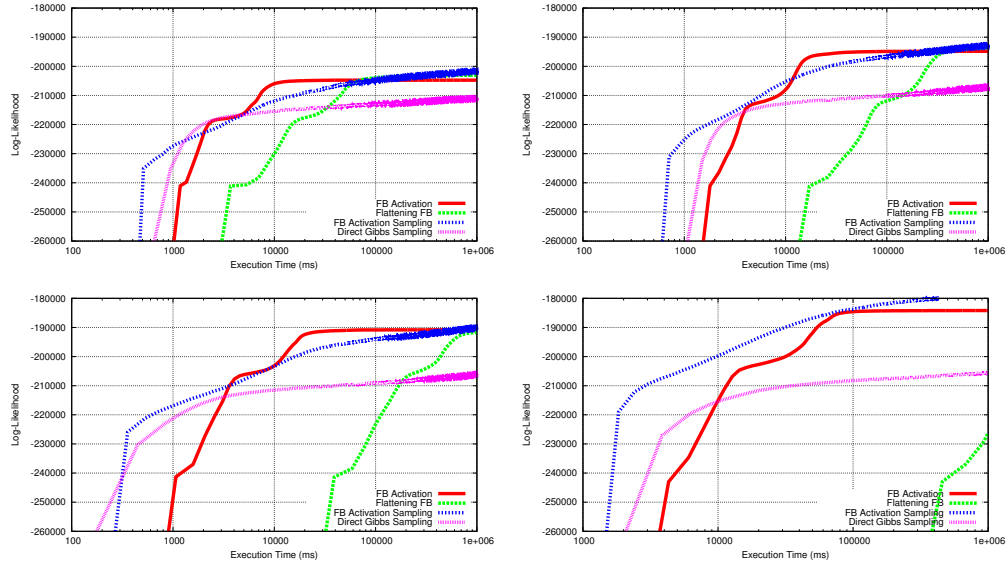

Figure 3: Convergence of log-likelihood for the training data on the Reuters corpus. Log-likelihood (vertical) is shown against the log-scaled execution time (horizontal) to display the execution time necessary to converge the learning of each algorithm. **(top-left)** HHMM of $D = 3$, $N = 3$. **(top-right)** $D = 3$, $N = 4$. **(bottom-left)** $D = 4$, $N = 3$. **(bottom-right)** HHMM of $D = 4$, $N = 4$.

Table 3: Average execution time for a single iteration (ms).

| Method | $D = 3, N = 3$ $(N^D = 27)$ | $D = 3, N = 4$ $(N^D = 64)$ | $D = 4, N = 3$ $(N^D = 81)$ | $D = 4, N = 4$ $(N^D = 256)$ |
|---|---|---|---|---|
| FBA | 186.65 | 391.73 | 476.92 | 1652.03 |
| FFB | 1729.90 | 9242.35 | 19257.80 | 220224.00 |
| FBAS | 82.45 | 142.20 | 183.39 | 581.58 |
| DGS | 24.19 | 37.50 | 45.43 | 265.98 |

The results show that the likelihood convergence using DGS is much slower than that of other methods. The execution time of DGS is less than that of other methods for a single iteration, but this cannot compensate for the low convergence speed. However, FBAS achieves a competitive likelihood in comparison to FBA. Results show that FBAS might be appropriate for some situations because FBAS finds a better solution than that FBA do in some results.

## 6 Conclusion

In this work, we proposed a new inference algorithm for HHMMs based on the activation probability. Results show that the performance of our proposed algorithm surpasses that of existing methods. The forward-backward activation algorithm described herein enables us to conduct unsupervised parameter learning with a practical computational cost for HHMMs of larger state space size.

## References

[1] C. Bishop. *Pattern Recognition and Machine Learning*. Springer, 2007.

[2] S. Chib. Calculating posterior distributions and modal estimates in markov mixture models. *Journal of Econometrics*, 1996.

[3] S. Fine, Y. Singer, and N. Tishby. The hierarchical hidden markov model: Analysis and applications. *Machine Learning*, 1998.

[4] T. Griffiths and M. Steyvers. Finding scientific topics. *Proc. the National Academy of Sciences of the United States of America*, 2004.

[5] K. Heller, Y. Teh, and D. Gorur. Infinite hierarchical hidden markov models. In *Proc. International Conference on Artificial Intelligence and Statistics*, 2009.

[6] S. Luhr, H. Bui, S. Venkatesh, and G. West. Recognition of human activity through hierarchical stochastic learning. In *Proc. Pervasive Computing and Communication*, 2003.

[7] K. Murphy and M. Paskin. Linear time inference in hierarchical hmms. In *Proc. Neural Information Processing Systems*, 2001.

[8] N. Nguyen, D. Phung, and S. Venkatesh. Learning and detecting activities from movement trajectories using the hierarchical hidden markov models. In *Proc. Computer Vision and Pattern Recognition*, 2005.

[9] L. Rabiner. A tutorial on hidden markov models and selected applications in speech recognition. *Proc. IEEE*, 1989.

[10] S. Scott. Bayesian methods for hidden markov models: Recursive computing in the 21st century. *Journal of the American Statistical Association*, 2002.

[11] M. Skounakis, M. Craven, and S. Ray. Hierarchical hidden markov models for information extraction. In *Proc. International Joint Conference on Artificial Intelligence*, 2003.

[12] M. Weiland, A. Smaill, and P. Nelson. Learning musical pitch structures with hierarchical hidden markov models. In *Proc. Journees Informatiques Musicales*, 2005.

[13] L. Xie, S. Chang, A. Divakaran, and H. Sun. Learning hierarchical hidden markov models for video structure discovery. Technical report, Columbia University, 2002.

